# Transfer Learning using Kolmogorov Complexity: Basic Theory and Empirical Evaluations

**M. M. Hassan Mahmud**
Department of Computer Science
University of Illinois at Urbana-Champaign
mmmahmud@uiuc.edu

**Sylvian R. Ray**
Department of Computer Science
University of Illinois at Urbana-Champaign
ray@cs.uiuc.edu

## Abstract

In transfer learning we aim to solve new problems using fewer examples using information gained from solving related problems. Transfer learning has been successful in practice, and extensive PAC analysis of these methods has been developed. However it is not yet clear how to define relatedness between tasks. This is considered as a major problem as it is conceptually troubling and it makes it unclear how much information to transfer and when and how to transfer it. In this paper we propose to measure the amount of information one task contains about another using conditional Kolmogorov complexity between the tasks. We show how existing theory neatly solves the problem of measuring relatedness and transferring the 'right' amount of information in sequential transfer learning in a Bayesian setting. The theory also suggests that, in a very formal and precise sense, no other reasonable transfer method can do much better than our Kolmogorov Complexity theoretic transfer method, and that sequential transfer is always justified. We also develop a practical approximation to the method and use it to transfer information between 8 arbitrarily chosen databases from the UCI ML repository.

## 1 Introduction

The goal of transfer learning [1] is to learn new tasks with fewer examples given information gained from solving related tasks, with each task corresponding to the distribution/probability measure generating the samples for that task. The study of transfer is motivated by the fact that people use knowledge gained from previously solved, related problems to solve new problems quicker. Transfer learning methods have been successful in practice, for instance it has been used to recognize related parts of a visual scene in robot navigation tasks, predict rewards in related regions in reinforcement learning based robot navigation problems, and predict results of related medical tests for the same group of patients. Figure 1 shows a prototypical transfer method [1], and it illustrates some of the key ideas. The $m$ tasks being learned are defined on the same input space, and are related by virtue of requiring the same common 'high level features' encoded in the hidden units. The tasks are learned in parallel – i.e. during training, the network is trained by alternating training samples from the different tasks, and the hope is that now the common high level features will be learned quicker. Transfer can also be done sequentially where information from tasks learned previously are used to speed up learning of new ones.

Despite the practical successes, the key question of how one measures relatedness between tasks has, so far, eluded answer. Most current methods, including the deep PAC theoretic analysis in [2], start by assuming that the tasks are related because they have a common near-optimal inductive bias (the common hidden units in the above example). As no explicit measure of relatedness is prescribed, it becomes difficult to answer questions such as how much information to transfer between tasks and when not to transfer information.

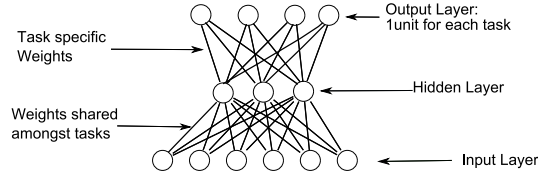

Figure 1: A typical Transfer Learning Method.

There has been some work which attempt to solve these problems. [3] gives a more explicit measure of task relatedness in which two tasks $P$ and $Q$ are said to be similar with respect to a given set of functions if the set contains an element $f$ such that $P(a) = Q(f(a))$ for all events $a$. By assuming the existence of these functions, the authors are able to derive PAC sample complexity bounds for error of each task (as opposed to expected error, w.r.t. a distribution over the $m$ tasks, in [2]). More interesting is the approach in [4], where the author derives PAC bounds in which the sample complexity is proportional to the joint *Kolmogorov complexity* [5] of the $m$ hypotheses. So Kolmogorov complexity (see below) determines the relatedness between tasks. However, the bounds hold only for $\geq 8192$ tasks (Theorem 3).

In this paper we approach the above idea from a Bayesian perspective and measure tasks relatedness using *conditional* Kolmogorov complexity of the hypothesis. We describe the basics of the theory to show how it justifies this approach and neatly solves the problem of measuring task relatedness (details in [6; 7]). We then perform experiments to show the effectiveness of this method.

Let us take a brief look at our approach. We assume that each hypothesis is represented by a program – for example a decision tree is represented by a program that contains a data structure representing the tree, and the relevant code to compute the leaf node corresponding to a given input vector. The Kolmogorov complexity of a hypothesis $h$ (or any other bit string) is now defined as the length of the shortest program that outputs $h$ given no input. This is a measure of absolute information content of an *individual object* – in this case the hypothesis $h$. It can be shown that Kolmogorov complexity is a sharper version of Information Theoretic entropy, which measures the amount of information in an *ensemble of objects* with respect to a *distribution* over the ensemble. The conditional Kolmogorov complexity of hypothesis $h$ given $h'$, $K(h|h')$, is defined as the length of the shortest program that outputs the program $h$ given $h'$ as input. $K(h|h')$ measures the amount of *constructive* information $h'$ contains about $h$ – how much information $h'$ contains for the purpose of constructing $h$. This is precisely what we wish to measure in transfer learning. Hence this becomes our measure of relatedness for performing sequential transfer learning in the Bayesian setting.

In the Bayesian setting, any sequential transfer learning mechanism/algorithm is 'just' a conditional prior $W(\cdot|h')$ over the hypothesis/probability measure space, where $h'$ is the task learned previously – i.e. the task we are trying to transfer information from. In this case, by setting the prior over the hypothesis space to be $P(\cdot|h') := 2^{-K(\cdot|h')}$ we weight each candidate hypothesis by how related it is to previous tasks, and so we automatically transfer the right amount of information when learning the new problem. We show that in a certain precise sense this prior is never much worse than any *reasonable* transfer learning prior, or any non-transfer prior. So, sequential transfer learning is always justified from a theoretical perspective. This result is quite unexpected as the current belief in the transfer learning community is that it should hurt to transfer from unrelated tasks. Due to lack of space, we only just briefly note that similar results hold for an appropriate interpretation of parallel transfer, and that, translated to the Bayesian setting, current practical transfer methods look like sequential transfer methods [6; 7]. Kolmogorov complexity is computable only in the limit (i.e. with infinite resources), and so, while ideal for investigating transfer in the limit, in practice we need to use an approximation of it (see [8] for a good example of this). In this paper we perform transfer in Bayesian decision trees by using a fairly simple approximation to the $2^{-K(\cdot|\cdot)}$ prior.

In the rest of the paper we proceed as follows. In section 3 we define Kolmogorov complexity more precisely and state all the relevant Bayesian convergence results for making the claims above. We then describe our Kolmogorov complexity based Bayesian transfer learning method. In section 4 we describe our method for approximation of the above using Bayesian decision trees, and then in section 5 we describe 12 transfer experiments using 8 standard databases from the UCI machine learning repository [9]. Our experiments are the most general that we know of, in the sense that we

transfer between arbitrary databases with little or no semantic relationships. We note that this fact also makes it difficult to compare our method to other existing methods (see also section 6).

## 2  Preliminaries

We consider Bayesian transfer learning for finite input spaces $\mathcal{I}_i$ and finite output spaces $\mathcal{O}_i$. We assume finite hypothesis spaces $\mathcal{H}_i$, where each $h \in \mathcal{H}_i$ is a conditional probability measure on $\mathcal{O}_i$, conditioned on elements of $\mathcal{I}_i$. So for $y \in \mathcal{O}_i$ and $x \in \mathcal{I}_i$, $h(y|x)$ gives the probability of output being $y$ given input $x$. Given $D_n = \{(x_1, y_1), (x_2, y_2), \cdots, (x_n, y_n)\}$ from $\mathcal{I}_i \times \mathcal{O}_i$, the probability of $D_n$ according to $h \in \mathcal{H}_i$ is given by:

$$h(D_n) := \prod_{k=1}^{n} h(y_k|x_k)$$

The conditional probability of a new sample $(x_{new}, y_{new}) \in \mathcal{I}_i \times \mathcal{O}_i$ for any conditional probability measure $\mu$ (e.g. $h \in \mathcal{H}_i$ or $M_W$ in ( 3.2) ) is given by:

$$\mu(y_{new}|x_{new}, D_n) := \frac{\mu(D_n \cup \{(x_{new}, y_{new})\})}{\mu(D_n)} \tag{2.1}$$

So the learning problem is: given a training sample $D_n$, where for each $(x_k, y_k) \in D_n$ $y_k$ is assumed to have been chosen according a $h \in \mathcal{H}_i$, learn $h$. The prediction problem is to predict the label of the new sample $x_{new}$ using ( 2.1). The probabilities for the inputs $x$ are not included above because they cancel out. This is merely the standard Bayesian setting, translated to a typical Machine learning setting (e.g. [10]).

We use MCMC simulations in a computer to sample for our Bayesian learners, and so considering only finite spaces above is acceptable. However, the theory we present here holds for any hypothesis, input and output space that may be handled by a computer with infinite resources (see [11; 12] for more precise descriptions). Note that we are considering cross-domain transfer [13] as our standard setting (see section 6). We further assume that each $h \in \mathcal{H}_i$ is a program (therefore a bit string) for some Universal prefix Turing machine $U$. When it is clear that a particular symbol $p$ denotes a program, we will write $p(x)$ to denote $U(p, x)$, i.e. running program $p$ on input $x$.

## 3  Transfer Learning using Kolmogorov Complexity

### 3.1  Kolmogorov Complexity based Task Relatedness

A program is a bit string, and a measure of absolute *constructive* information that a bit string $x$ contains about another bit string $y$ is given by the conditional Kolmogorov complexity of $x$ given $y$ [5] . Since our hypotheses are programs/bit strings, the amount of information that a hypothesis or program $h'$ contains about constructing another hypothesis $h$ is also given by the same:

**Definition 1.** *The conditional Kolmogorov complexity of $h \in \mathcal{H}_j$ given $h' \in \mathcal{H}_i$ is defined as the length of the shortest program that given the program $h'$ as input, outputs the program $h$.*

$$K(h|h') := \min_r \{l(r) : r(h') = h\}$$

We will use a minimality property of $K$. Let $f(x, y)$ be a computable function over product of bit strings. $f$ is computable means that there is a program $p$ such that $p(x, n)$, $n \in \mathbb{N}$, computes $f(x)$ to accuracy $\epsilon < 2^{-n}$ in finite time. Now assume that $f(x, y)$ satisfies for each $y$ $\sum_x 2^{-f(x,y)} \leq 1$. Then for a constant $c_f = K(f) + O(1)$, *independent of $x$ and $y$*, but *dependent on $K(f)$*, the length of shortest program computing $f$, and some small constant ($O(1)$) [5, Corollary 4.3.1]:

$$K(x|y) \leq f(x, y) + c_f \tag{3.1}$$

### 3.2  Bayesian Convergence Results

A Bayes mixture $M_W$ over $\mathcal{H}_i$ is defined as follows:

$$M_W(D_n) := \sum_{h \in \mathcal{H}_i} h(D_n)W(h) \text{ with } \sum_{h \in \mathcal{H}_i} W(h) \leq 1 \tag{3.2}$$

(the inequality is sufficient for the convergence results). Now assume that the data has been generated by a $h_j \in \mathcal{H}_i$ (this is standard for a Bayesian setting, but we will relax this constraint below). Then the following impressive result holds true for each $(x, y) \in \mathcal{I}_i \times \mathcal{O}_i$.

$$\sum_{t=0}^{\infty} \sum_{D_n} h_j(D_n)[M_W(y|x, D_n) - h_j(y|x, D_n)]^2 \leq -\ln W(h_j). \qquad (3.3)$$

So for finite $-\ln W(h_j)$, convergence is rapid; the expected number of times $n \, |M_W(a|x, D_n) - h_j(a|x, D_n)| > \epsilon$ is $\leq -\ln W(h_j)/\epsilon^2$, and the probability that the number of $\epsilon$ deviations $> -\ln W(h_j)/\epsilon^2 \delta$ is $< \delta$. This result was first proved in [14], and extended variously in [11; 12]. In essence these results hold as long as $\mathcal{H}_i$ can be enumerated and $h_j$ and $W$ can be computed with infinite resources. These results also hold if $h_j \notin \mathcal{H}_i$, but $\exists h'_j \in \mathcal{H}_i$ such that the $n^{th}$ order KL divergence between $h_j$ and $h'_j$ is bounded by $k$. In this case the error bound is $-\ln W(h'_j) + k$ [11, section 2.5]. Now consider the Solomonoff-Levin prior: $2^{-K(h)}$ – this has ( 3.3) error bound $K(h) \ln 2$, and for any computable prior $W(\cdot)$, $f(x, y) := -\ln W(x)/\ln 2$ satisfies conditions for $f(x, y)$ in ( 3.1). So by ( 3.3), with $y =$ the empty string, we get:

$$K(h) \ln 2 \leq -\ln W(h) + c_W \qquad (3.4)$$

By ( 3.3), this means that for all $h \in \mathcal{H}_i$, the error bound for the $2^{-K(h)}$ prior can be no more than a constant worse than the error bound for any other prior. Since reasonable priors have small $K(W)$ $(= O(1))$, $c_W = O(1)$ and this prior is *universally optimal* [11, section 5.3].

## 3.3 Bayesian Transfer Learning

Assume we have previously observed/learned $m - 1$ tasks, with task $t_j \in \mathcal{H}_{i_j}$, and the $m^{th}$ task to be learned is in $\mathcal{H}_{i_m}$. Let $\mathbf{t} := (t_1, t_2, \cdots, t_{m-1})$. In the Bayesian framework, a transfer learning scheme corresponds to a computable prior $W(\cdot|\mathbf{t})$ over the space $\mathcal{H}_{i_m}$,

$$\sum_{h \in \mathcal{H}_{i_m}} W(h|\mathbf{t}) \leq 1$$

In this case, by ( 3.3), the error bound of the transfer learning scheme $M_W$ (defined by the prior $W$) is $-\ln W(h|\mathbf{t})$. We define our transfer learning method $M_{TL}$ by choosing the prior $2^{-K(\cdot|\mathbf{t})}$:

$$M_{TL}(D_n) := \sum_{h \in \mathcal{H}_{i_m}} h(D_n) 2^{-K(h|\mathbf{t})}.$$

For $M_{TL}$ the error bound is $K(h|\mathbf{t}) \ln 2$. By the minimality property ( 3.1), we get that

$$K(h|\mathbf{t}) \ln 2 \leq -\ln W(h|\mathbf{t}) + c_W$$

So for a reasonable computable transfer learning scheme $M_W$, $c_W = O(1)$ and for all $h$ and $\mathbf{t}$, the error bound for $M_{TL}$ is no more than a constant worse than the error bound for $M_W$ – i.e. $M_{TL}$ is *universally optimal* [11, section 5.3]. Also note that in general $K(x|y) \leq K(x)$[1]. Therefore by ( 3.4) the transfer learning scheme $M_{TL}$ is also universally optimal over all non-transfer learning schemes – i.e. in the precise formal sense of the framework in this paper, sequential transfer learning is always justified. The result in this section, while novel, are not *technically deep* (see also [6] [12, section 6]). We should also note that the $2^{-K(h)}$ prior is *not* universally optimal with respect to the transfer prior $W(\cdot|\mathbf{t})$ because the inequality ( 3.4) now holds only upto the constant $c_{W(\cdot|\mathbf{t})}$ which depends on $K(\mathbf{t})$. So this constant increases with increasing number of tasks which is very undesirable. Indeed, this is demonstrated in our experiments when the base classifier used is an approximation to the $2^{-K(h)}$ prior and the error of this prior is seen to be significantly higher than the transfer learning prior $2^{-K(h|\mathbf{t})}$.

## 4 Practical Approximation using Decision Trees

Since $K$ is computable only in the limit, to apply the above ideas in practical situations, we need to approximate $K$ and hence $M_{TL}$. Furthermore we also need to specify the spaces $\mathcal{H}_i, \mathcal{O}_i, \mathcal{I}_i$ and how to sample from the approximation of $M_{TL}$. We address each issue in turn.

## 4.1 Decision Trees

We will consider standard binary decision trees as our hypotheses. Each hypothesis space $\mathcal{H}_i$ consists of decision trees for $\mathcal{I}_i$ defined by the set $\mathbf{f}_i$ of features. A tree $h \in \mathcal{H}_i$ is defined recursively:

$$h := \mathbf{n}_{root}$$

$$\mathbf{n}_j := r_j \ \mathbf{C}_j \ \emptyset \ \emptyset \mid r_j \ \mathbf{C}_j \ \mathbf{n}_L^j \ \emptyset \mid r_j \ \mathbf{C}_j \ \emptyset \ \mathbf{n}_R^j \mid r_j \ \mathbf{C}_j \ \mathbf{n}_L^j \ \mathbf{n}_R^j$$

$\mathbf{C}$ is a vector of size $|\mathcal{O}_i|$, with component $\mathbf{C}_i$ giving the probability of the $i^{th}$ class. Each rule $r$ is of the form $f < v$, where $f \in \mathbf{f}_i$ and $v$ is a value for $f$. The vector $\mathbf{C}$ is used during classification only when the corresponding node has one or more $\emptyset$ children. The size of each tree is $Nc_0$ where $N$ is the number of nodes, and $c_0$ is a constant, denoting the size of each rule entry, the outgoing pointers, and $\mathbf{C}$. Since $c_0$ and the length of the program code $p_0$ for computing the tree output are constants independent of the tree, we define the length of a tree as $l(h) := N$.

## 4.2 Approximating $K$ and the Prior $2^{-K(\cdot|\mathbf{t})}$

*Approximation for a single previously learned tree:* We will approximate $K(\cdot|\cdot)$ using a function that is defined for a single previously learned tree as follows:

$$C_{ld}(h|h') := l(h) - d(h, h')$$

where $d(h, h')$ is the maximum number of overlapping nodes starting from the root nodes:

$$d(h, h') := d(\mathbf{n}_{root}, \mathbf{n}'_{root}) \qquad\qquad d(\mathbf{n}, \emptyset) := 0$$

$$d(\mathbf{n}, \mathbf{n}') := 1 + d(\mathbf{n}_L, \mathbf{n}'_L) + d(\mathbf{n}_R, \mathbf{n}'_R) \qquad\qquad d(\emptyset, \mathbf{n}') := 0$$

In the single task case, the prior is just $2^{-l(h)}/Z_l$ (which is an approximation to the Solomonoff-Levin prior $2^{-K(\cdot)}$), and in the transfer learning case, the prior is $2^{-C_{ld}(\cdot|h')}/Z_{C_{ld}}$ where the $Z$s are normalization terms[2]. In both cases, we can sample from the prior directly by growing the decision tree dynamically. Call a $\emptyset$ in $h$ a hole. Then for $2^{-l(h)}$, during the generation process, we first generate an integer $k$ according to $2^{-t}$ distribution (easy to do using a pseudo random number generator). Then at each step we select a hole uniformly at random and then create a node there (with two more holes) and generate the corresponding rule randomly. We do so until we get a tree with $l(h) = k$. In the transfer learning case, for the prior $2^{-C_{ld}(\cdot|h')}$ we first generate an integer $k$ according to $2^{-t}$ distribution. Then we generate as above until we get a tree $h$ with $C_{ld}(h|h') = k$. It can be seen with a little thought that these procedures sample from the respective priors.

*Approximation for multiple previously learned trees:* We define $C_{ld}$ for multiple trees as an averaging of the contributions of each of the $m - 1$ previously learned trees:

$$C_{ld}^m(h_m|h_1, h_2, \cdots, h_{m-1}) := -\log\left(\frac{1}{m-1} \sum_{i=1}^{m-1} 2^{-C_{ld}(h_m|h_i)}\right)$$

In the transfer learning case, we need to sample according $2^{-C_{ld}^m(\cdot|\cdot)}/Z_{C_{ld}^m}$ which reduces to $1/[(m-1)Z_{C_{ld}^m}]\sum_{i=1}^{m-1} 2^{-C_{ld}(h_m|h_i)}$. To sample from this, we can simply select a $h_i$ from the $m - 1$ trees at random and then sample from $2^{-C_{ld}(\cdot|h_i)}$ to get the new tree.

*The transfer learning mixture:* The approximation of the transfer learning mixture $M_{TL}$ is now:

$$P_{TL}(D_n) = \sum_{h \in \mathcal{H}_{i_m}} h(D_n) 2^{-C_{ld}^m(h|\mathbf{t})}/Z_{C_{ld}^m}$$

So by ( 3.3), the error bound for $P_{TL}$ is given by $C_{ld}^m(h|\mathbf{t})\ln 2 + \ln Z_{C_{ld}}$ (the $\ln Z_{C_{ld}}$ is a constant that is same for all $h \in \mathcal{H}_i$). So when using $C_{ld}^m$, universality is maintained, but only up to the degree that $C_{ld}^m$ approximates $K$. In our experiments we used the prior $1.005^{-C}$ instead of $2^{-C}$ above to make larger trees more likely and hence speed up convergence of MCMC sampling.

Table 1: Metropolis-Hastings Algorithm

1. Let $D_n$ be the training sample; select the current tree/state $h_{cur}$ using the proposal distribution $q(h_{cur})$.
2. For $i = 1$ to $J$ do
   (a) Choose a candidate next state $h_{prop}$ according to the proposal distribution $q(h_{prop})$.
   (b) Draw $u$ uniformly at random from $[0, 1]$ and set $h_{cur} := h_{prop}$ if $A(h_{prop}, h_{cur}) > u$, where $A$ is defined by

   $$A(h, h') := \min \left\{ 1, \frac{h(D_n) 2^{-C_{ld}^m(h|\mathbf{t})} q(h')}{h'(D_n) 2^{-C_{ld}^m(h'|\mathbf{t})} q(h)} \right\}$$

## 4.3 Approximating $P_{TL}$ using Metropolis-Hastings

As in standard Bayesian MCMC methods, the idea will be to draw $N$ samples $h_{m_i}$ from the posterior, $P(h|D_n, \mathbf{t})$ which is given by

$$P(h|D_n, \mathbf{t}) := h(D_n) 2^{-C_{ld}^m(h|\mathbf{t})} / (Z_{C_{ld}^m} P(D_n))$$

Then we will approximate $P_{TL}$ by

$$\hat{P}_{TL}(y|x) := \frac{1}{N} \sum_{i=1}^{N} h_{m_i}(y|x)$$

We will use the standard Metropolis-Hastings algorithm to sample from $P_{TL}$ (see [15] for a brief introduction and further references). The algorithm is given in table 1. The algorithm is first run for some $J = T$, to get the Markov chain $q \times A$ to converge, and then starting from the last $h_{cur}$ in the run, the algorithm is run again for $J = N$ times to get $N$ samples for $\hat{P}_{TL}$. In our experiments we set $T$ to 1000 and $N = 50$. We set $q$ to our prior $2^{-C_{ld}^m(\cdot|\mathbf{t})} / Z_{C_{ld}^m}$, and hence the acceptance probability $A$ is reduced to $\min\{1, h(D_n)/h'(D_n)\}$. Note that every time after we generate a tree according to $q$, we set the $\mathbf{C}$ entries using the training sample $D_n$ in the usual way.

## 5 Experiments

We used 8 databases from the UCI machine learning repository [9] in our experiments (table 2). To show transfer of information we used 20% of the data for a task as the training sample, but also used as prior knowledge trees learned on another task using 80% of the data as training sample. The reported error rates are on the testing sets and are averages over 10 runs . To the best of our knowledge our transfer experiments are the most general performed so far, in the sense that the databases information is transferred between have semantic relationship that is often tenuous.

We performed 3 sets of experiments. In the first set we learned each classifier using 80% of the data as training sample and 20% as testing sample (since it is a Bayesian method, we did not use a validation sample-set). This set ensured that our base Bayesian classifier with $2^{-l(h)}$ prior is reasonably powerful and that any improvement in performance in the transfer experiments (set 3) was due to transfer and not deficiency in our base classifier. From a survey of literature it seems the error rate for our classifier is always at least a couple of percentage points better than C4.5. As an example, for *ecoli* our classifier outperforms Adaboost and Random Forests in [16], but is a bit worse than these for *German Credit*.

In the second set of experiments we learned the databases that we are going to transfer to using 20% of the database as training sample, and 80% of the data as the testing sample. This was done to establish baseline performance for the transfer learning case. The third and final set of experiments were performed to do the actual transfer. In this case, first one task was learned using 80/20 (80% training, 20% testing) data set and then this was used to learn a 20/80 dataset. During transfer, the $N$ trees from the sampling of the 80/20 task were all used in the prior $2^{-C_{ld}^N(\cdot|\mathbf{t})}$. The results are

Table 2: Database summary. The last column gives the error and standard deviation for 80/20 database split.

| Data Set | No. of Samples | No. of Feats. | No. Classes | Error/S.D. |
|---|---|---|---|---|
| Ecoli | 336 | 7 | 8 | 9.8%, 3.48 |
| Yeast | 1484 | 8 | 10 | 14.8%, 2.0 |
| Mushroom | 8124 | 22 | 2 | 0.83%, 0.71 |
| Australian Credit | 690 | 14 | 2 | 16.6%, 3.75 |
| German Credit | 1000 | 20 | 2 | 28.2%, 4.5 |
| Hepatitis | 155 | 19 | 2 | 18.86%, 2.03 |
| Breast Cancer,Wisc. | 699 | 9 | 2 | 5.6%, 1.9 |
| Heart Disease, Cleve. | 303 | 14 | 5 | 23.0%, 2.56 |

given in table 3. In our experiments, we transferred only to tasks that showed a significant drop in error rate with the $20/80$ split. Surprisingly, the error of the other data sets did not change much.

As can be seen from comparing the tables, in most cases transfer of information improves the performance compared to the baseline transfer case. For *ecoli*, the transfer resulted in improvement to near $80/20$ levels, while for *australian* the improvement was better than $80/20$. While the error rate for *mushroom* and *bc-wisc* did not move up to $80/20$ levels, there was improvement. Interestingly transfer learning did not hurt in one single case, which agrees with our theoretical results in the idealized setting.

Table 3: Results of 12 transfer experiments. *Transfer To* and *From* rows gives databases information is transferred to and from. The row *No-Transfer* gives the baseline $20/80$ error-rate and standard deviation. Row *Transfer* gives the error rate and standard deviation after transfer, and the final row *PI* gives percentage improvement in performance due to transfer. With our admittedly inefficient code, each experiment took between $15 - 60$ seconds on a 2.4 GHz laptop with 512 MB RAM.

| Trans. To | ecoli | | | Australian | | |
|---|---|---|---|---|---|---|
| Trans. From | *Yeast* | *Germ.* | *BC Wisc* | *Germ.* | *ecoli* | *hep.* |
| No-Transfer | 20.6%, 3.8 | 20.6%, 3.8 | 20.6%, 3.8 | 23.2%, 2.4 | 23.2%, 2.4 | 23.2%, 2.4 |
| Transfer | 11.3%, 1.6 | 10.2%, 4.74 | 9.68%, 2.98 | 15.47%, 0.67 | 15.43%, 1.2 | 15.21%, 0.42 |
| PI | 45.1% | 49% | 53% | 33.0% | 33.5% | 34.4% |

| Trans. To | mushroom | | | BC Wisc. | | |
|---|---|---|---|---|---|---|
| Trans. From | *ecoli* | *BC Wisc.* | *Germ.* | *heart* | *Aus.* | *ecoli* |
| No-Transfer | 13.8%, 1.3 | 13.8%, 1.3 | 13.8%, 1.3 | 10.3%, 1.6 | 10.3%, 1.6 | 10.3%, 1.6 |
| Transfer | 4.6%, 0.17 | 4.64%, 0.21 | 3.89%, 1.02 | 8.3%, 0.93 | 8.1%, 1.22 | 7.8%, 2.03 |
| PI | 66.0% | 66.0% | 71.8% | 19.4% | 21.3% | 24.3% |

# 6 Discussion

In this paper we introduced a Kolmogorov Complexity theoretic framework for Transfer Learning. The theory is universally optimal and elegant, and we showed its practical applicability by constructing approximations to it to transfer information across disparate domains in standard UCI machine learning databases. The full theoretical development can be found in [6; 7]. Directions for future empirical investigations are many. We did not consider transferring from multiple previous tasks, and effect of size of source samples on transfer performance (using $70/30$ etc. as the sources) or transfer in regression. Due to the general nature of our method, we can perform transfer experiments between any combination of databases in the UCI repository. We

also wish to perform experiments using more powerful generalized similarity functions like the gzip compressor [8][3].

We also hope that it is clear that Kolmogorov complexity based approach elegantly solves the problem of cross-domain transfer, where we transfer information between tasks that are defined over different input,output and distribution spaces. To the best of our knowledge, the first paper to address this was [13], and recent works include [17] and [18]. All these methods transfer information by finding structural similarity between various networks/rule that form the hypotheses. This is, of course, a way to measure constructive similarity between the hypotheses, and hence an approximation to Kolmogorov complexity based similarity. So Kolmogorov complexity elegantly unifies these ideas. Additionally, the above methods, particularly the last two, are rather elaborate and are hypothesis space specific ([18] is even task specific). The theory of Kolmogorov complexity and its practical approximations such as [8] and this paper suggests that we can get good performance by just using generalized compressors, such as gzip, etc., to measure similarity.

### Acknowledgments

We would like to thank Kiran Lakkaraju for their comments and Samarth Swarup for many fruitful disucssions.

## Footnotes

[1]Because $\arg K(x)$, with a constant length modification, also outputs $x$ given input $y$.

[2]The $Z$'s exist, here because the $\mathcal{H}$s are finite, and in general because $k_i = Nc_0 + l(p_0)$ gives lengths of programs, which are known to satisfy $\sum_i 2^{-k_i} \leq 1$.

[3]A flavor of this approach: if the standard compressor is gzip, then the function $C_{gzip}(xy)$ will give the length of the string $xy$ after compression by gzip. $C_{gzip}(xy) - C_{gzip}(y)$ will be the conditional $C_{gzip}(x|y)$. So $C_{gzip}(h|h')$ will give the relatedness between tasks.

### References

[1] Rich Caruana. Multitask learning. *Machine Learning*, 28:41–75, 1997.

[2] Jonathan Baxter. A model of inductive bias learning. *Journal of Artificial Intelligence Research*, 12:149–198, March 2000.

[3] Shai Ben-David and Reba Schuller. Exploiting task relatedness for learning multiple tasks. In *Proceedings of the $16^{th}$ Annual Conference on Learning Theory*, 2003.

[4] Brendan Juba. Estimating relatedness via data compression. In *Proceedings of the $23^{rd}$ International Conference on Machine Learning*, 2006.

[5] Ming Li and Paul Vitanyi. *An Introduction to Kolmogorov Complexity and its Applications*. Springer-Verlag, New York, 2nd edition, 1997.

[6] M. M. Hassan Mahmud. On universal transfer learning. In *Proceedings of the $18^{th}$ International Conference on Algorithmic Learning Theory*, 2007.

[7] M. M. Hassan Mahmud. On universal transfer learning (Under Review). 2008.

[8] R. Cilibrasi and P. Vitanyi. Clustering by compression. *IEEE Transactions on Information theory*, 51(4):1523–1545, 2004.

[9] D.J. Newman, S. Hettich, C.L. Blake, and C.J. Merz. UCI repository of ML databases, 1998.

[10] Radford M. Neal. Bayesian methods for machine learning, NIPS tutorial, 2004.

[11] Marcus Hutter. Optimality of Bayesian universal prediction for general loss and alphabet. *Journal of Machine Learning Research*, 4:971–1000, 2003.

[12] Marcus Hutter. On universal prediction and bayesian confirmation. *Theoretical Computer Science (in press)*, 2007.

[13] Samarth Swarup and Sylvian R. Ray. Cross domain knowledge transfer using structured representations. In *Proceedings of the $21^{st}$ National Conference on Artificial Intelligence (AAAI)*, 2006.

[14] R. J. Solomonoff. Complexity-based induction systems: comparisons and convergence theorems. *IEEE Transactions on Information Theory*, 24(4):422–432, 1978.

[15] Christophe Andrieu, Nando de Freitas, Arnaud Doucet, and Michael I. Jordan. An introduction to MCMC for machine learning. *Machine Learning*, 50(1-2):5–43, 2003.

[16] Leo Breiman. Random forests. *Machine Learning*, 45:5–32, 2001.

[17] Lilyana Mihalkova, Tuyen Huynh, and Raymond Mooney. Mapping and revising markov logic networks for transfer learning. In *Proceedings of the $22^{nd}$ National Conference on Artificial Intelligence (AAAI*, 2007.

[18] Matthew Taylor and Peter Stone. Cross-domain transfer for reinforcement learning. In *Proceedings of the $24^{th}$ International Conference on Machine Learning*, 2007.

